# Dopamine Bonuses

**Sham Kakade**          **Peter Dayan**
Gatsby Computational Neuroscience Unit
17 Queen Square, London, England, WC1N 3AR.
sham@gatsby.ucl.ac.uk    dayan@gatsby.ucl.ac.uk

## Abstract

Substantial data support a temporal difference (TD) model of dopamine (DA) neuron activity in which the cells provide a global error signal for reinforcement learning. However, in certain circumstances, DA activity seems anomalous under the TD model, responding to non-rewarding stimuli. We address these anomalies by suggesting that DA cells multiplex information about reward bonuses, including Sutton's exploration bonuses and Ng *et al*'s non-distorting shaping bonuses. We interpret this additional role for DA in terms of the unconditional attentional and psychomotor effects of dopamine, having the computational role of guiding exploration.

## 1  Introduction

Much evidence suggests that dopamine cells in the primate midbrain play an important role in reward and action learning. Electrophysiological studies support a theory that DA cells signal a global prediction error for summed future reward in appetitive conditioning tasks (Montague *et al*, 1996; Schultz *et al*, 1997), in the form of a temporal difference prediction error term. This term can simultaneously be used to train predictions (in the model, the projections of the DA cells in the ventral tegmental area to the limbic system and the ventral striatum) and to train actions (the projections of DA cells in the substantia nigra to the dorsal striatum and motor and premotor cortex). Appetitive prediction learning is associated with classical conditioning, the task of learning which stimuli are associated with reward; appetitive action learning is associated with instrumental conditioning, the task of learning actions that result in reward delivery.

The computational role of dopamine in reward learning is controversial for two main reasons (Ikemoto & Panksepp, 1999; Redgrave *et al*, 1999). First, stimuli that are not associated with reward prediction are known to activate the dopamine system persistently, including in particular stimuli that are novel and salient, or that physically resemble other stimuli that do predict reward (Schultz, 1998). Second, dopamine release is associated with a set of motor effects, such as species- and stimulus-specific approach behaviors, that seem either irrelevant or detrimental to the delivery of reward. We call these unconditional effects.

In this paper, we study this apparently anomalous activation of the DA system, suggesting that it *multiplexes* information about bonuses, potentially including exploration bonuses (Sutton, 1990; Dayan & Sejnowski, 1996) and shaping bonuses (Ng *et al*, 1999), on top of reward prediction errors. These responses are associated with the unconditional effects of DA, and are part of an attentional system.

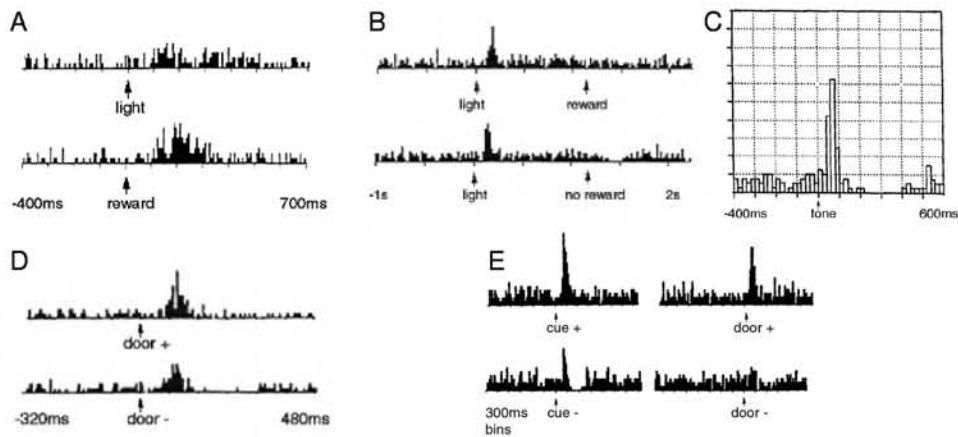

Figure 1: Activity of individual DA neurons — though substantial data suggest the homogeneous character of these responses (Schultz, 1998). See text for description. The latency and duration of the DA activation is about 100ms. The depression has duration of about 200 ms. The baseline spike rate is about 2-4 Hz. Adapted from Schultz *et al* (1990, 1992, & 1993) and Jacobs *et al* (1997).

## 2   DA Activity

Figure 1 shows three different types of dopamine responses that have been observed by Schultz *et al* and Jacobs *et al*. Figures 1A;B show the response to a conditioned stimulus that becomes predictive of reward (CS+). For this, in early trials (figure 1A), there is no, or only a weak response to the CS+, but a strong response just after the time of delivery of the reward. In later trials (figure 1B), after learning is complete (but before overtraining), the DA cells are activated in response to the stimulus, and fire at background rates to the reward. Indeed, if the reward is omitted, there is depression of DA activity at just the time during early trials that it used to excite the cells. These are the key data for which the temporal difference model accounts. Under the model, the cells report the temporal difference (TD) error for reward, *ie* the difference in amount of reward that is delivered and the amount that is expected. Let $r(t)$ be the amount of reward received at time $t$ and $v(t)$ be the prediction of the sum total (undiscounted) reward to be delivered in a trial after time $t$, or:

$$v(t) \sim \sum_{\tau \geq 0} r(\tau + t) \,. \tag{1}$$

The TD component to the dopamine activity is the prediction error:

$$\delta(t) = r(t) + v(t+1) - v(t) \tag{2}$$

which uses $r(t) + v(t+1)$ as an estimate of $\Sigma_{\tau \geq 0} r(\tau + t)$, so that the TD error is an estimate of $\Sigma_{\tau \geq 0} r(\tau + t) - v(t)$. Provided that the information about state includes information about how much time has elapsed since the CS+ was presented (which must be available because of the precisely timed nature of the inhibition at the time of reward, if the expected reward is not presented), this model accounts well for the results in figure 1A.

The general framework of reinforcement learning methods for Markov decision problems (MDPs) extends these results to the case of control. An MDP consists of states, actions, transition probabilities between states under the chosen action,

and the associated rewards with these transitions. The goal of the subject solving a MDP is to find a policy (a choice of actions in each state) so as to optimize the sum total reward it receives. The TD error $\delta(t)$ can be used to learn optimal policies by implementing a form of policy iteration, which is an optimal control technique that is standard in engineering (Sutton & Barto, 1998; Bertsekas & Tsitsiklis, 1996).

Figures 1C;D show that reporting a prediction error for reward does not exhaust the behavioral repertoire of the DA cells. Figure 1C shows responses to salient, novel, stimuli. The dominant effect is that there is a phasic activation of dopamine cells followed by a phasic inhibition, both locked to the stimulus. These novelty responses decrease over trials, but quite slowly for very salient stimuli (Schultz, 1998). In some cases, particularly in early trials of appetitive learning (figure 1A top), there seems to be little or no phasic inhibition of the cells following the activation. Figure 1D shows what happens when a stimulus (door $-$) that resembles a reward-predicting stimulus (door $+$) is presented without reinforcement. Again a phasic increase over baseline followed by a depression is seen (lower 1D). However, unlike the case in figure 1B, there is no persistent reward prediction, since if a reward is subsequently delivered (unexpectedly), the cells become active (not shown) (Schultz, 1998).

## 3   Multiplexing and reward distortion

The most critical issue is whether it is possible to reconcile the behavior of the DA cells seen in figures 1C;D with the putative computational role of DA in terms of reporting prediction error for reward. Intuitively, these apparently anomalous responses are *benign*, that is they do not interfere with the end point of normal reward learning, provided that they sum to zero over a trial.

To see this, consider what happens once learning is complete. If we sum the prediction error terms from equation 2, starting from the time of the stimulus onset at $t = 1$, we get

$$\sum_{t \geq 1} \delta(t) = v(t_{\text{end}}) - v(1) + \sum_{t \geq 1} r(t)$$

where $t_{\text{end}}$ is the time at the end of the trial. Assuming that $v(t_{\text{end}}) = 0$ and $v(1) = 0$, *ie* that the monkey confines its reward predictions to within a trial, we can see that any additional influences on $\delta(t)$ that sum to 0 preserve predicted sum future rewards. From figure 1, this seems true of the majority of the extra responses, *ie* anomalous activation is canceled by anomalous inhibition, though it is not true of the uncancelled DA responses shown in figure 1A (upper). Altogether, DA activity can still be used to learn predictions and choose actions – although it should not strictly be referred to solely in terms of prediction error for reward.

Apart from the issue of anomalous activation that is *not* canceled (upper figure 1A), this leaves open two key questions: what drives the extra DA responses; and what effects do they have. We offer a set of possible interpretations (mostly associated with *bonuses*) that it is hard to decide between on the basis of current data.

## 4   Novelty and Bonuses

Three very different sorts of bonuses have been considered in reinforcement learning, novelty, shaping and exploration bonuses. The presence of the first two of these is suggested by the responses in figure 1. Bonuses modify the reward signals and so change the course of learning. They are mostly used to guide exploration of the world, and are typically heuristic ways of addressing the computationally intractable exploration-exploitation dilemma.

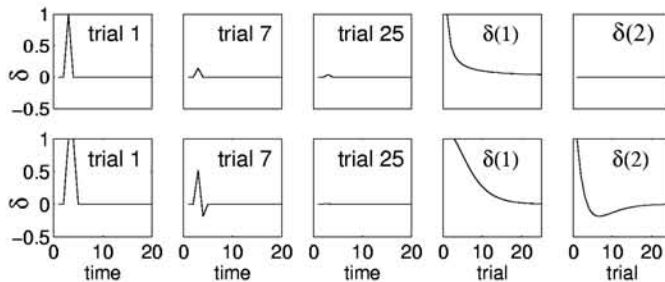

Figure 2: Activity of the DA system given novelty bonuses. The plots show different aspects of the TD error $\delta$ as a function of time $t$ within a trial (first three plots in each row) or as a function of number $T$ of trials (last two). Upper) A novelty signal was applied for just the first timesteps of the stimulus and decayed hyperbolically with trial number as $1/T$. Lower) A novelty signal was applied for the first two timesteps of the stimulus and now decayed exponentially as $e^{-.3T}$ to demonstrate that the precise form of decay is irrelevant. Trial numbers and times are shown in the plots. The learning rate was $\epsilon = 0.3$.

We first consider a *novelty bonus*, which we take as a model for uncancelled anomalous activity. A novelty bonus is a value that is added to states or state-action pairs associated with their unfamiliarity — novelty is made intrinsically rewarding. This is computationally reasonable, at least in moderation, and indeed it has become standard practice in reinforcement learning to use optimistic initial values for states to encourage systems to plan to get to novel or unfamiliar states. In TD terms, this is like replacing the true environmental reward $r(t)$ at time $t$ with

$$r(t) \rightarrow r(t) + n(\mathbf{x}(t), T)$$

where $\mathbf{x}(t)$ is the state at time $t$ and $n(\mathbf{x}(t), T)$ is the novelty of this state in trial $T$ (an index we generally suppress). The effect on the TD error is then

$$\delta(t) = r(t) + n(\mathbf{x}(t), T) + v(t+1) - v(t) \tag{3}$$

The upper plots in figure 2 show the effect of including such an exploration bonus, in a case in which just the first timestep of a new stimulus in any given trial are awarded a novelty signal which decays hyperbolically to 0 as the stimulus becomes more familiar. Here, a novel stimulus is presented for a 25 trials without there being any reward consequences. The effect is just a positive signal which decreases over time. Learning has no effect on this, since the stimulus cannot predict away a novelty signal that lasts only a single timestep. The lower plots in figure 2 show that it is possible to get partial apparent cancellation through learning, if the novelty signal is applied for the first two timesteps of a stimulus (for instance if the novelty signal is calculated relatively slowly). In this case, the initial effect is just a positive signal (leftmost graph), the effect of TD learning gives it a negative transient after a few trials (second plot), and then, as the novelty signal decays to 0, the effect goes away (third plot). The righthand plots show how $\delta(t)$ behaves across trials. If there was no learning, then there would be no negative transient. The depression of the DA signal comes from the decay of the novelty bonuses.

Novelty bonuses are true bonuses in the sense that they actually distort the reward function. In particular, this means that we would not expect the sum of the extra TD error terms to be 0 across a trial. This property makes them useful, for instance, in actually distorting the optimal policy in Markov decision problems to ensure that exploration is planned and executed in favor of exploitation. However, they can be dangerous for exactly the same reason – and there are reports of them leading to incorrect behavior, making agents search too much.

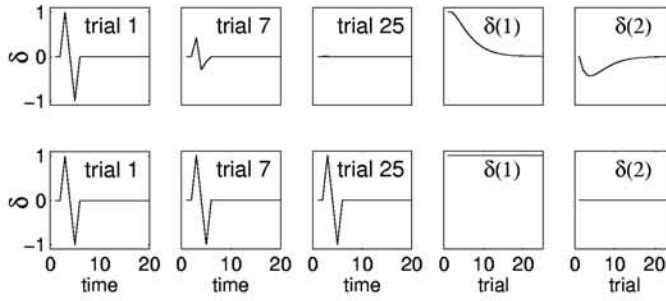

Figure 3: Activity of the DA system given shaping bonuses (in the same format as figure 2). Upper) The plots show different aspects of the TD error $\delta$ as a function of time $t$ within a trial (first three plots) or as a function of number $T$ of trials (last two). Here, the shaping bonus comes from a $\phi(t) = 0$ for the first two timesteps a stimulus is presented within a trial (t=1;2), and 0 thereafter, irrespective of trial number. The learning rate was $\epsilon = 0.3$. Lower) The same plots for $\epsilon = 0$

In answer to this concern, Ng *et al* (1999) invented the idea of non-distorting *shaping* bonuses. Ng *et al*'s shaping bonuses are guaranteed *not* to distort optimal policies, although they can still change the exploratory behavior of agents. This guarantee comes because a shaping bonus is derived from a potential function $\phi(\mathbf{x})$ of a state, distorting the TD error to

$$\delta(t) = r(t) + \phi(\mathbf{x}(t+1)) - \phi(\mathbf{x}(t)) + v(t+1) - v(t) \qquad (4)$$

The difference from the novelty bonus of equation 3 is that the bonus comes from the *difference* between the potential functions for one state and the previous state, and they thus cancel themselves out when summed over a trial. Shaping bonuses must remain constant for the guarantee about the policies to hold.

The upper plots in figure 3 show the effect of shaping bonuses on the TD error. Here, the potential function is set to the value 1 for the first two timesteps of a stimulus in a trial, and 0 otherwise. The most significant difference between shaping and novelty bonuses is that the former exhibits a negative transient even in the very first trial, whereas, for the latter, it is a learned effect. If the learning rate is non-zero, then shaping bonuses are exactly predicted away over the course of normal learning. Thus, even though the same bonus is provided on trial 25 as trial 1, the TD error becomes 0 since the shaping bonus is predicted away. The dynamics of the decay shown in the last two plots is controlled by the learning rate for TD. The lower plots show what happens if learning is switched off at the time the shaping bonus is provided – this would be the case if the system responsible for computing the bonus takes its effect before the inputs associated with the stimulus are plastic. In this case, the shaping bonus is preserved.

The final category of bonus is an ongoing *exploration* bonus (Sutton, 1990; Dayan & Sejnowski, 1996) which is used to ensure continued exploration. Sutton (1990) suggested adding to the estimated value of each state (or each state-action pair), a number proportional to the length of time since it was last visited. This ultimately makes it irresistible to go and visit states that have not been visited for a long time. Dayan & Sejnowski (1996) derived a bonus of this form from a model of environmental change that justifies the bonus. There is no evidence for this sort of continuing exploration bonus in the dopamine data, perhaps not surprisingly, since the tasks undertaken by the monkey offer little possibility for any trade-off between exploration and exploitation.

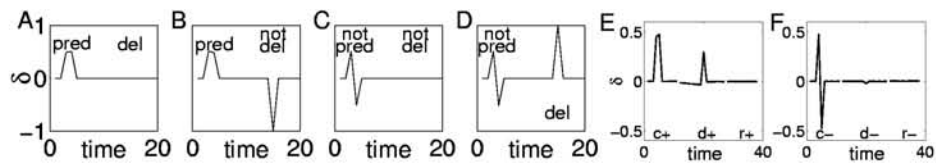

Figure 4: Activity $\delta(t)$ of the dopamine for partial predictability. del = delivered, pred = predicted. A;B) CS+ is presented with (A) or surprisingly, without (B) reward. C;D) CS- is presented without (C) or surprisingly, with (D) reward. On each trial, an initial stimulus (presented at $t = 3$ is ambiguous as to whether CS+ or CS- is presented (each occurs equally often), and the ambiguity is perfectly resolved at $t = 4$. E;F) The model shows the same behavior. Since the CS± comes at a random interval after the cue, the traces are stimulus locked to the relevant events.

## 5   Generalization Responses and Partial Observability

Generalization responses (figure 1D) show a persistent effect of stimuli that merely resemble a rewarded stimulus. However, animals do not terminally confuse normally rewarded and normally non-rewarded stimuli, since if a reward is provided in the latter case, then it engenders DA activity (as an unexpected reward should), and if it is not provided, then there is no depression (as would be the case if an expected reward was not delivered) (Schultz, 1998).

One possibility is that this activity comes from a shaping bonus that is not learned away, as in the lower plots of figure 3. An alternative interpretation comes from partial observability. If the initial information from the world is ambiguous as to whether the stimulus is actually rewarding (door+, called CS+ trials) or non-rewarding (door-, called CS- trials), because of the similarity, then the animal should develop an initial expectation that there could be a reward (whose mean value is related to the degree of confusion). This should lead to a partial activation of the DA system. If the expectation is canceled by subsequent information about the stimulus (available, for instance, following a saccade), then the DA system will be inhibited below baseline exactly to nullify the earlier positive prediction. If the expectation is confirmed, then there will be continued activity representing the difference between the value of the reward and the expected value given the ambiguous stimulus. Figure 4 shows an example of this in a simplified case for which the animal receives information about the true stimulus over two timesteps, the first timestep is ambiguous to the tune of 50%; the second perfectly resolves the ambiguity. Figures 4A;B show CS+ trials, with and without the delivery of reward; figures 4C;D CS- trials, without and with the delivery of reward. The similarity of 4A;C to figure 1D is clear.

Another instance of this generalization response is shown in figure 1E. Here, an cue light (c±) is provided indicating whether a CS+ or a CS- (d±) is to appear at a random later time, which in turn is followed (or not) after a fixed interval by a reward (r±). DA cells show a generalization response to the cue light; and then fire to the CS+ or are unaffected by the CS-; and finally do not respond to the appropriate presence or absence of the cue. Figures 4E;F shows that this is exactly the behavior of the model. The DA response stimulus locked to CS+ arises because of the variability in the interval between the cue light and the CS+; if this interval were fixed, then the cells would only respond to the cue (c+), as in Schultz (1993).

## 6   Discussion

We have suggested a set of interpretations for the activity of the DA system to add to that of reporting prediction error for reward. The two theoretically most

interesting features are novelty and shaping bonuses. The former distort the reward function in such a way to encourage exploration of new stimuli, and new places. The latter are non-distorting, and can be seen as being multiplexed by the DA system together with the prediction error signal.

Since shaping bonuses are not distorting they have no ultimate effect on action choice. However, the signal provided by the activation (and then cancellation) of DA can nevertheless have a significant neural effect. We suggest that DA release has *unconditional* effects in the ventral striatum (perhaps allowing stimuli to be read into pre-frontal working memory, Cohen *et al*, 1998) and the dorsal striatum (perhaps engaging stimulus-directed approach and exploratory orienting behaviors, see Ikemoto & Panksepp (1999) for review). For stimuli that actually predict rewards (and so cause an initial activation of the DA system), these behaviors are often called appetitive; for novel, salient, and potentially important stimuli that are not known to predict rewards, they allow the system to pay appropriate attention. These effects of DA are unconditional, since they are hard-wired and not learned. In the case of partial observability, DA release due to the uncertain prediction of reward directly causes further investigation, and therefore resolution of the uncertainty. When unconditional and conditioned behaviors conflict, the former seem to dominate, as in the inability of animals to learn to run *away* from a stimulus in order to get food from it.

The most major lacuna in the model is its lack of one or more opponent processes to DA that might report on punishments and the absence of predicted rewards. There is substantial circumstantial evidence that this might be one role for serotonin (which itself has unconditional effects associated with fear, fight, and flight responses that are opposite to those of DA), but there is not the physiological evidence to support or refute this possibility. Understanding the interaction of dopamine and serotonin in terms of their conditioned and unconditioned effects is a major task for future work.

**Acknowledgements**

Funding is from the NSF and the Gatsby Charitable Foundation.

# References

[1] Bertsekas, DP & Tsitsitklis, JN (1996). *Neuro-dynamic Programming.* Cambridge, MA: Athena Scientific.

[2] Cohen, JD, Braver, TS & O'Reilly, RC (1998). In AC Roberts, TW Robbins, editors, *The Prefrontal Cortex: Executive and Cognitive Functions.* Oxford: OUP.

[3] Dayan, P, & Sejnowski, TJ (1996). *Machine Learning,* **25**: 5-22.

[4] Horvitz, JC, Stewart, T, & Jacobs, B, (1997). *Brain Research,* **759**:251-258.

[5] Ikemoto, S, & Panksepp, J, (1999). *Brain Research Reviews,* **31**:6-41.

[6] Montague, PR, Dayan, P, & Sejnowski, TJ, (1996). *Journal of Neuroscience,* **16**:1936-1947.

[7] Ng, AY, Harada, D, and Russell, S, (1999). *Proceedings of the Sixteenth International Conference on Machine Learning.*

[8] Redgrave, P, Prescott, T, & Gurney, K (1999). *Trends in Neurosciences,* **22**: 146-151.

[9] Schultz, W, (1992). *Seminars in the Neurosciences,* **4**: 129-138.

[10] Schultz, W, (1998). *Journal of Neurophysiology,* **80**: 1-27.

[11] Schultz, W, Apicella, P, & Ljungberg, T, (1993). *Journal of Neuroscience,* **13**: 900-913.

[12] Schultz, W, Dayan, P, and Montague, PR, (1997). *Science,* **275**: 1593-1599.

[13] Schultz, W, & Romo, R, (1990). *Journal of Neuroscience,* **63**: 607-624.

[14] Sutton, RS, (1990). *Machine Learning: Proceedings of the Seventh International Conference,* 216-224.

[15] Sutton, RS & Barto, AG (1998). *Reinforcement Learning: An Introduction.* Cambridge, MA: MIT Press.
